# Learning Global Direct Inverse Kinematics

**David DeMers***
Computer Science & Eng.
UC San Diego
La Jolla, CA 92093-0114

**Kenneth Kreutz-Delgado[†]**
Electrical & Computer Eng.
UC San Diego
La Jolla, CA 92093-0407

## Abstract

We introduce and demonstrate a bootstrap method for construction of an inverse function for the robot kinematic mapping using only sample configuration–space/workspace data. Unsupervised learning (clustering) techniques are used on pre–image neighborhoods in order to *learn* to partition the configuration space into subsets over which the kinematic mapping is invertible. Supervised learning is then used separately on each of the partitions to approximate the inverse function. The ill–posed inverse kinematics function is thereby regularized, and a global inverse kinematics solution for the wristless Puma manipulator is developed.

## 1 INTRODUCTION

The robot forward kinematics function is a continuous mapping

$$f : \mathcal{C} \subseteq \Theta^n \to \mathcal{W} \subseteq \mathcal{X}^m$$

which maps a set of $n$ joint parameters from the *configuration space*, $\mathcal{C}$, to the $m$–dimensional *task space*, $\mathcal{W}$. If $m \leq n$, the robot has *redundant* degrees–of–freedom (dof's). In general, control objectives such as the positioning and orienting of the end–effector are specified with respect to task space co–ordinates; however, the manipulator is typically controlled only in the configuration space. Therefore, it is important to be able to find some $\vec{\theta} \in \mathcal{C}$ such that f($\vec{\theta}$) is a particular target value $\vec{x}_0 \in \mathcal{W}$. This is the *inverse kinematics problem*.

[†]e-mail: kreutz@ece.ucsd.edu

The inverse kinematics problem is ill–posed. If there are redundant dof's then the problem is *locally* ill–posed, because the solution is non–unique and consists of a non–trivial manifold[1] in $C$. With or without redundant dof's, the problem is generally *globally* ill–posed because of the existence of a finite set of solution branches — there will typically be multiple configurations which result in the same task space location. Thus computation of a *direct* inverse is problematic due to the many–to–one nature (and therefore non–invertibility) of the map $f$[2].

The inverse problem can be solved explicitly, that is, in closed form, for only certain kinds of manipulators. E.g. six dof elbow manipulators with separable wrist (where the first three joints are used for positioning and the last three have a common origin and are used for orientation), such as the Puma 560, are solvable, see (Craig, 86). The alternative to a closed form solution is a numerical solution, usually either using the inverse of the Jacobian , which is a Newton-style approach, or by using gradient descent (also a Jacobian–based method). These methods are iterative and require expensive Jacobian or gradient computation at each step, thus they are not well–suited for real–time control.

Neural networks can be used to find an inverse by implementing either direct inverse modeling (estimating the explicit function $f^{-1}$) or differential methods. Implementations of the direct inverse approach typically fail due to the non–linearity of the solution set[3], or resolve this problem by restriction to a single solution *a priori*. However, such a prior restriction of the solutions may not be possible or acceptable in all circumstances, and may drastically reduce the dexterity and manipulability of the arm.

The differential approaches either find only the nearest local solution, or resolve the multiplicity of solutions at training time, as with Jordan's forward modeling (Jordan & Rumelhart, 1990) or the approach of (Nguyen & Patel, 1990). We seek to regularize the mapping in such a way that all possible solutions are available at run–time, and can be computed efficiently as a direct constant–time inverse rather than approximated by slower iterative differential methods. To achieve the fast run–time solution, a significant cost in training time must be paid; however, it is not unreasonable to invest resources in off–line learning in order to attain on–line advantages. Thus we wish to gain the run–time computational efficiency of a direct inverse solution while also achieving the benefits of the differential approaches.

This paper introduces a method for performing *global* regularization; that is, identifying the complete, finite set of solutions to the inverse kinematics problem for a non–redundant manipulator. This will provide the ability to *choose* a particular solution at run time. Resolving redundancy is beyond the scope of this paper; however, preliminary work on a method which may be integrated with the work presented here is shown in (DeMers & Kreutz-Delgado, 1991). In the remainder of this paper it will be assumed that the manipulator does not have redundant dof's. It will also be assumed that all of the joints are revolute, thus the configuration space is a subset of the $n$-torus, $T^n$.

# 2  TOPOLOGY OF THE KINEMATICS FUNCTION

The kinematics mapping is continuous and smooth and, generically, neighborhoods in configuration space map to neighborhoods in the task space[4]. The configuration space, $\mathcal{C}$, is made up of a finite number of disjoint regions or partitions, separated by $n-1$ dimensional surfaces where the Jacobian loses rank (called *critical* surfaces), see (Burdick, 1988, Burdick, 1991).

Let $f : T^n \to \mathbf{R}^n$ be the kinematic mapping. Then

$$W = f(\mathcal{C}) = \bigcup_{i=1}^{k} f_i(\mathcal{C}_i)$$

where $f_i$ is the restriction of $f$ to $\mathcal{C}_i$, $f_i : \mathcal{C}_i = \Theta^n / f \to \mathbf{R}^n$ and the factor space $\Theta^n / f$ is locally diffeomorphic to $\mathbf{R}^n$. The $\mathcal{C}_i$ are each a connected region such that

$$\forall \vec{\theta} \in \mathcal{C}_i, \quad \det\left(J(\vec{\theta})\right) \neq 0$$

where J is the Jacobian of $f$, $J = d_\theta f$. Define $\mathcal{W}_i$ as $f(\mathcal{C}_i)$. Generically, $f_i$ is one–to–one and onto open neighborhoods of $\mathcal{W}_i$[5], thus by the inverse function theorem

$$\exists\, g_i(\vec{x}) = f_i^{-1} : \mathcal{W}_i \to \mathcal{C}_i, \text{ such that } f \circ g_i(\vec{x}) = \vec{x}, \quad \forall \vec{x} \in \mathcal{W}_i$$

In the general case, with redundant dof's, the kinematics over a single configuration–space region can be viewed as a fiber bundle, where the fibers are homeomorphic to $T^{n-m}$. The base space is the reachable workspace (the image of $\mathcal{C}_i$ under $f$). Solution branch resolution can be done by identifying distinct connected open coordinate neighborhoods of the configuration space which cover the workspace. Redundancy resolution can be done by a consistent parameterization of the fibers within each neighborhood. In the case at hand, without redundant dof's, the "fibers" are singleton sets and no resolution is needed.

In the remainder of this paper, we will use input/output data to identify the individual regions, $\mathcal{C}_i$, of a non–redundant manipulator, over which the mapping $f_i : \mathcal{C}_i \to \mathcal{W}_i$ is invertible. The input/output data will then be partitioned modulo the configuration regions $\mathcal{C}_i$, and each $f_i^{-1}$ approximated individually.

# 3  SAMPLING APPROACH

If the manipulator can be measured and a large sample of $(\vec{\theta}, \vec{x})$ pairs taken, stored such that the $\vec{x}$ samples can be searched efficiently, a rough estimate of the inverse solutions at a particular target point $\vec{x}_0$ may be obtained by finding all of the $\vec{\theta}$ points whose image lies within some $\epsilon$ of $\vec{x}_0$. The pre–image of this $\epsilon$–ball will generically consist of several distinct (distorted) balls in the configuration space. If the sampling is adequate then there will be one such ball for each of the inverse solution branches. If each of the the points in each ball is given a label for the solution branch, the labeled data may then be used for supervised

learning of a classifier of solution branches in the configuration space. In this way we will have "bootstrapped" our way to the development of a solution branch classifier.

Taking advantage of the continuous nature of the forward mapping, note that if $\vec{x}_0$ is slightly perturbed by a "jump" to a neighboring target point then the pre–image balls will also be perturbed. We can assign labels to the new data consistent with labels already assigned to the previous data, by computing the distances between the new, unlabeled balls and the previously labeled balls. Continuing in this fashion, $\vec{x}_0$ traces a path through the entire workspace and solution branch labels may be given to all points in $C$ which map to within $\epsilon$ of one of the selected $\vec{x}$ points along the sweep.

This procedure results in a significant and representative proportion of the data now being labeled as to solution branch. Thus we now have labeled data $(\vec{\theta}, \vec{x}, B(\vec{\theta}))$, where $B(\vec{\theta}) = \{1, \ldots, k\}$ indicates which of the $k$ solution branches, $C_i$, the point $\vec{\theta}$ is in. We can now construct a classifier using supervised learning to compute the branches $B(\vec{\theta})$ for a given $\vec{\theta}$. Once an estimate of $B(\vec{\theta})$ is developed, we may use it to classify large amounts of $(\vec{\theta}, \vec{x})$ data, and partition the data into $k$ sets, one for each of the solution branches, $C_i$.

## 4   RESOLUTION OF SOLUTION BRANCHES

We applied the above to the wristless Puma 560, a 3–R manipulator for end–effector positioning in $\mathbf{R}^3$. We took 40,000 samples of $(\vec{\theta}, \vec{x})$ points, and examined all points within 10 cm of selected target values $\vec{x}_i$. The $\vec{x}_i$ formed a grid of 90 locations in the workspace. 3,062 of the samples fell within 10 cm of one of the $\vec{x}_i$. The configuration space points for each target $\vec{x}_i$ were clustered into four groups, corresponding to the four possible solution branches of the wristless Puma 560. About 3% of the points were clustered into the wrong group, based on the labeling scheme used. These 3,062 points were then used as training patterns for a feedforward neural network classifier. A point was classified into the group associated with the output unit of the neural network with maximum activation. The output values were normalized to sum to 1.0. The network was tested on 50,000 new, previously unseen $(\vec{\theta}, \vec{x})$ pairs, and correctly classified more than 98% of them.

All of the erroneous classifications were for points near the critical surfaces. Therefore the activation levels of the output units can be used to estimate closeness to a critical surface. Examining the test data and assigning all $\vec{\theta}$ points for which no output unit has activation greater than or equal to 0.8 to the "near–a–singularity" class, the remaining points were 100% correctly classified.

Figure 1 shows the true critical manifold separating the regions of configuration space, and the estimated manifold consisting of points from the test set where the maximum activation of output units of the trained neural network is less than 0.8. The configuration space is a subset of the 3–torus, which is shown here "sliced" along three generators and represented as a cube. Because the Puma 560 has physical limits on the range of motion of its joints, the regions shown are in fact six distinct regions, and there is no wraparound in any direction. This classifier network is our candidate for an estimate of $B(\vec{\theta})$. With it, the samples can be separated into groups corresponding to the domains of each of the $f_i$, thus regularizing into $k = 6$ one–to–one invertible pieces[6].

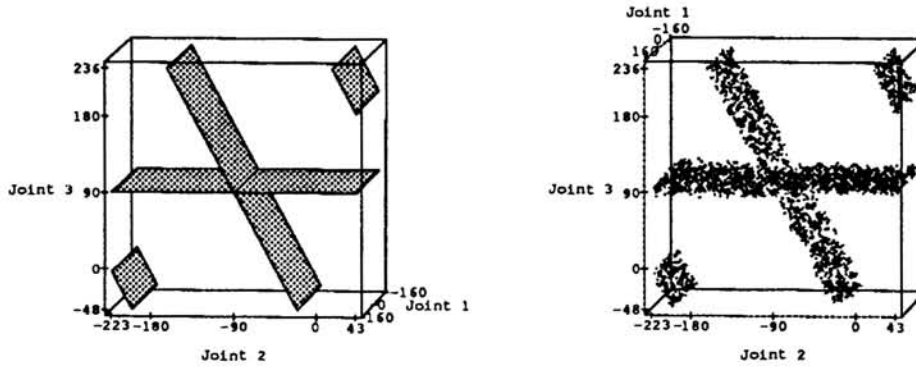

Figure 1: *The analytically derived critical surfaces, along with 1,000 points for which no unit of the neural network classifier has greater than 0.8 activation.*

## 5   DIRECT INVERSE SOLUTIONS

The classifier neural network can now be used to partition the data into four groups, one for each of the branches, $C_i$. For each of these data sets, we train a feedforward network to learn the mapping in the inverse direction. The target vectors were represented as vectors of the sine of the half–angle (a measure motivated by the quaternion representation of orientation). MSE under 0.001 were achieved for each of the four. This looks like a very small error, however, this error is somewhat misleading. The configuration space error is measured in units which are difficult to interpret. More important is the error in the workspace when the solution computed is used in the forward kinematics mapping to position the arm. Over a test set of 4,000 points, the average positioning error was 5.2 cm over the 92 cm radius workspace.

We have as yet made no attempts to optimize the network or training for the direct inverse; the thrust of our work is in achieving the regularization. It is clear that substantially better performance can be developed, for example, by following (Ritter, et al., 1989), and we expect end–effector positioning errors of less than 1% to be easily achievable.

## 6   DISCUSSION

We have shown that by exploiting the topological property of continuity of the kinematic mapping for a non–redundant 3–dof robot we can determine all of the solution regions of the inverse kinematic mapping. We have mapped out the configuration space critical surfaces and thus discovered an important topological property of the mapping, corresponding to an important physical property of the manipulator, by unsupervised learning. We can boostrap from the original input/output data, unlabeled as to solution branch, and construct an accurate classifier for the entire configuration space. The data can thereby be partitioned into sets which are individually one–to–one and invertible, and the inverse mapping can be directly approximated for each. Thus a large learning–time investment results in a fast run–time direct inverse kinematics solution.

---

cube shown would be a true 3–torus, with opposite faces identified. Thus the small pieces in the corners would be part of the larger regions by wraparound in the Joint 2 direction.

We need a thorough sampling of the configuration space in order to ensure that enough points will fall within each $\epsilon$-ball, thus the data requirements are clearly exponential in the number of degrees of freedom of the manipulator. Even with efficient storage and retrieval in geometric data structures, such as a k–d tree, high dimensional systems may not be tractable by our methods.

Fortunately practical and useful robotic systems of six and seven degrees of freedom should be amenable to this method, especially if separable into positioning and orienting subsystems.

### Acknowledgements

This work was supported in part by NSF Presidential Young Investigator award IRI–9057631 and a NASA/Netrologic grant. The first author would like to thank NIPS for providing student travel grants. We thank Gary Cottrell for his many helpful comments and enthusiastic discussions.

## Footnotes

*e-mail: demers@cs.ucsd.edu

[1] Generically of dimensionality equal to $n - m$.

[2] The target values are assumed to be in the range of $f$, $\vec{x} \in \mathcal{W} = f(\mathcal{C})$, so the *existence* of a solution is not an issue in this paper.

[3] Training a network to minimize mean squared error with multiple target values for the same input value results in a "learned" response of the average of the targets. Since the targets lie on a number of non–linear manifolds (for the redundant case) or consist of a finite number of points (for the non–redundant case), the average of multiple targets will typically not be a correct target.

[4]This property fails when the manipulator is in a *singular* configuration, at which the Jacobian, $d_\theta f$, loses rank.

[5]Since it is generically true that J is non–singular.

[6]Although there are only four inverse solutions for any $\vec{x}$. If there were no joint limits, then the

### References

Joel Burdick (1991), "A Classification of 3R Regional Manipulator Singularities and Geometries", *Proc. 1991 IEEE Intl. Conf. Robotics & Automation*, Sacramento.

Joel Burdick (1988), "Kinematics and Design of Redundant Robot Manipulators", Stanford Ph.D. Thesis, Dept. of Mechanical Engineering.

John Craig (1986), *Introduction to Robotics*, Addison-Wesley.

David DeMers & Kenneth Kreutz–Delgado (1991), "Learning Global Topological Properties of Robot Kinematic Mappings for Neural Network-Based Configuration Control", in Bekey, ed. *Proc. USC Workshop on Neural Networks in Robotics*, (to appear).

Michael I. Jordan (1988), "Supervised Learning and Systems with Excess Degrees of Freedom", COINS Technical Report 88–27, University of Massachusetts at Amherst.

Michael I. Jordan & David E. Rumelhart (1990), "Forward Models: Supervised Learning with a Distal Teacher". Submitted to *Cognitive Science*.

L. Nguyen & R.V. Patel (1990), "A Neural Network Based Strategy for the Inverse Kinematics Problem in Robotics", in Jamshidi and Saif, eds., *Robotics and Manufacturing: recent Trends in Research, Education and Applications*, vol. 3, pp. 995–1000 (ASME Press).

Helge J. Ritter, Thomas M. Martinetz, & Klaus J. Schulten (1989), "Topology–Conserving Maps for Learning Visuo–Motor–Coordination", *Neural Networks*, Vol. 2, pp. 159–168.